# The Use of Dynamic Writing Information in a Connectionist On-Line Cursive Handwriting Recognition System

**Stefan Manke**      **Michael Finke**      **Alex Waibel**

University of Karlsruhe
Computer Science Department
D-76128 Karlsruhe, Germany
manke@ira.uka.de, finkem@ira.uka.de

Carnegie Mellon University
School of Computer Science
Pittsburgh, PA 15213-3890, U.S.A.
waibel@cs.cmu.edu

## Abstract

In this paper we present $\mathbf{NPen^{++}}$, a connectionist system for writer independent, large vocabulary on-line cursive handwriting recognition. This system combines a robust input representation, which preserves the dynamic writing information, with a neural network architecture, a so called Multi-State Time Delay Neural Network (MS-TDNN), which integrates recognition and segmentation in a single framework. Our preprocessing transforms the original coordinate sequence into a (still temporal) sequence of feature vectors, which combine strictly local features, like curvature or writing direction, with a bitmap-like representation of the coordinate's proximity. The MS-TDNN architecture is well suited for handling temporal sequences as provided by this input representation. Our system is tested both on writer dependent and writer independent tasks with vocabulary sizes ranging from 400 up to 20,000 words. For example, on a 20,000 word vocabulary we achieve word recognition rates up to 88.9% (writer dependent) and 84.1% (writer independent) without using any language models.

# 1  INTRODUCTION

Several preprocessing and recognition approaches for on-line handwriting recognition have been developed during the past years. The main advantage of on-line handwriting recognition in comparison to optical character recognition (OCR) is the temporal information of handwriting, which can be recorded and used for recognition. In general this dynamic writing information (i.e. the time-ordered sequence of coordinates) is not available in OCR, where input consists of scanned text. In this paper we present the **NPen**$^{++}$ system, which is designed to preserve the dynamic writing information as long as possible in the preprocessing and recognition process.

During preprocessing a temporal sequence of $N$-dimensional feature vectors is computed from the original coordinate sequence, which is recorded on the digitizer. These feature vectors combine strictly local features, like curvature and writing direction [4], with so-called context bitmaps, which are bitmap-like representations of a coordinate's proximity.

The recognition component of **NPen**$^{++}$ is well suited for handling temporal sequences of patterns, as provided by this kind of input representation. The recognizer, a so-called Multi-State Time Delay Neural Network (MS-TDNN), integrates recognition and segmentation of words into a single network architecture. The MS-TDNN, which was originally proposed for continuous speech recognition tasks [6, 7], combines shift-invariant, high accuracy pattern recognition capabilities of a TDNN [8, 4] with a non-linear alignment procedure for aligning strokes into character sequences.

Our system is applied both to different writer dependent and writer independent, large vocabulary handwriting recognition tasks with vocabulary sizes up to 20,000 words. Writer independent word recognition rates range from 92.9% with a 400 word vocabulary to 84.1% with a 20,000 word vocabulary. For the writer dependent system, word recognition rates for the same tasks range from 98.6% to 88.9% [1].

In the following section we give a description of our preprocessing performed on the raw coordinate sequence, provided by the digitizer. In section 3 the architecture and training of the recognizer is presented. A description of the experiments to evaluate the system and the results we have achieved on different tasks can be found in section 4. Conclusions and future work is described in section 5.

# 2  PREPROCESSING

The dynamic writing information, i.e. the temporal order of the data points, is preserved throughout all preprocessing steps. The original coordinate sequence $\{(\bar{x}(t), \bar{y}(t))\}_{t \in \{0 \ldots T'\}}$ recorded on the digitizer is transformed into a new temporal sequence $\boldsymbol{x}_0^T = \boldsymbol{x}_0 \ldots \boldsymbol{x}_T$, where each frame $\boldsymbol{x}_t$ consists of an $N$-dimensional real-valued feature vector $(f_1(t), \ldots, f_N(t)) \in [-1, 1]^N$.

Several normalization methods are applied to remove undesired variability from the original coordinate sequence. To compensate for different sampling rates and varying writing speeds the coordinates originally sampled to be equidistant in time are resampled yielding a new sequence $\{(\tilde{x}(t), \tilde{y}(t))\}_{t \in \{0 \ldots T\}}$ which is equidistant in

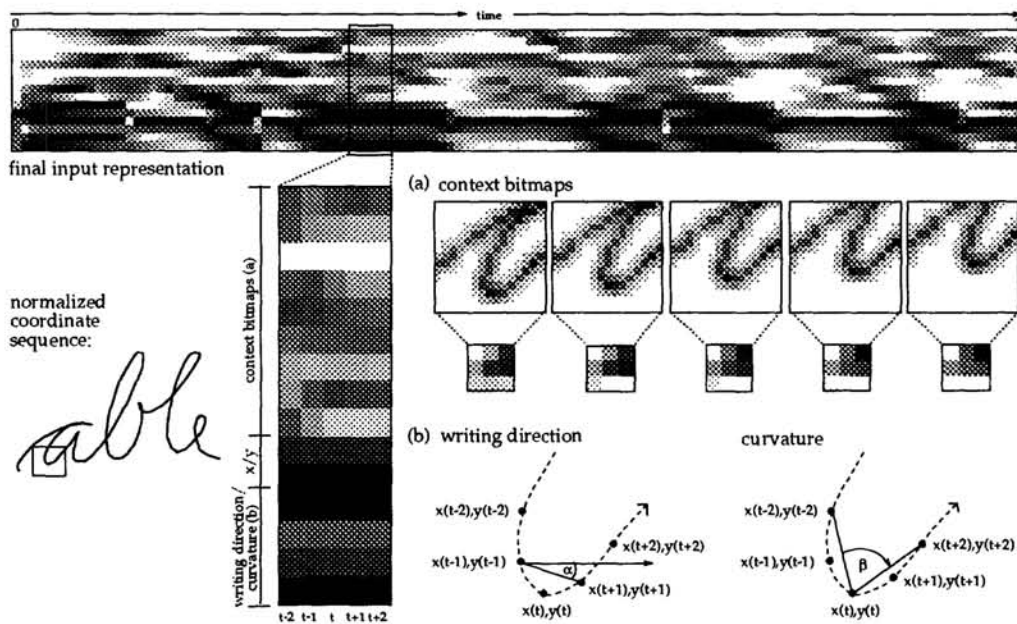

Figure 1: Feature extraction for the normalized word "able". The final input representation is derived by calculating a 15-dimensional feature vector for each data point, which consists of a context bitmap (a) and information about the curvature and writing direction (b).

space. This resampled trajectory is smoothed using a moving average window in order to remove sampling noise. In a final normalization step the goal is to find a representation of the trajectory that is reasonably invariant against rotation and scaling of the input. The idea is to determine the words' baseline using an EM approach similar to that described in [5] and rescale the word such that the center region of the word is assigned to a fixed size.

From the normalized coordinate sequence $\{(x(t), y(t))\}_{t \in \{0...T\}}$ the temporal sequence $x_0^T$ of $N$-dimensional feature vectors $x_t = (f_1(t), \ldots, f_N(t))$ is computed (Figure 1). Currently the system uses $N = 15$ features for each data point. The first two features $f_1(t) = x(t) - x(t-1)$ and $f_2(t) = y(t) - b$ describe the relative $X$ movement and the $Y$ position relative to the baseline $b$. The features $f_3(t)$ to $f_6(t)$ are used to describe the curvature and writing direction in the trajectory [4] (Figure 1(b)). Since all these features are strictly local in the sense that they are local both in time and in space they were shown to be inadequate for modeling temporal long range context dependencies typically observed in pen trajectories [2]. Therefore, nine additional features $f_7(t)$ to $f_{15}(t)$ representing $3 \times 3$ bitmaps were included in each feature vector (Figure 1(a)). These so-called context bitmaps are basically low resolution, bitmap-like descriptions of the coordinate's proximity, which were originally described in [2].

Thus, the input representation as shown in Figure 1 combines strictly local features like writing direction and curvature with the context bitmaps, which are still local

in space but global in time. That means, each point of the trajectory is visible from each other point of the trajectory in a small neighbourhood. By using these context bitmaps in addition to the local features, important information about other parts of the trajectory, which are in a limited neighbourhood of a coordinate, are encoded.

## 3 THE NPen$^{++}$ RECOGNIZER

The **NPen**$^{++}$ recognizer integrates recognition and segmentation of words into a single network architecture, the Multi-State Time Delay Neural Network (MS-TDNN). The MS-TDNN, which was originally proposed for continuous speech recognition tasks [6, 7], combines the high accuracy single character recognition capabilities of a TDNN [8, 4] with a non-linear time alignment algorithm (dynamic time warping) for finding stroke and character boundaries in isolated handwritten words.

### 3.1 MODELING ASSUMPTIONS

Let $W = \{w_1, \ldots w_K\}$ be a vocabulary consisting of $K$ words. Each of these words $w_i$ is represented as a sequence of characters $w_i \equiv c_{i_1} c_{i_2} \ldots c_{i_k}$ where each character $c_j$ itself is modelled by a three state hidden markov model $c_j \equiv q_j^0 q_j^1 q_j^2$. The idea of using three states per character is to model explicitly the initial, middle and final section of the characters. Thus, $w_i$ is modelled by a sequence of states $w_i \equiv q_{i_0} q_{i_1} \ldots q_{j_{3k}}$. In these word HMMs the self-loop probabilities $p(q_{i_j}|q_{i_j})$ and the transition probabilities $p(q_{i_j}|q_{i_{j-1}})$ are both defined to be $\frac{1}{2}$ while all other transition probabilities are set to zero.

During recognition of an unknown sequence of feature vectors $x_0^T = x_0 \ldots x_T$ we have to find the word $w_i \in W$ in the dictionary that maximizes the a-posteriori probability $p(w_i|x_0^T, \theta)$ given a fixed set of parameters $\theta$ and the observed coordinate sequence. That means, a written word will be recognized such that

$$w_j = \text{argmax}_{w_i \in W} p(w_i|x_0^T, \theta).$$

In our Multi-State Time Delay Neural Network approach the problem of modeling the word posterior probability $p(w_i|x_0^T, \theta)$ is simplified by using Bayes' rule which expresses that probability as

$$p(w_i|x_0^T, \theta) = \frac{p(x_0^T|w_i, \theta)P(w_i|\theta)}{p(x_0^T|\theta)}.$$

Instead of approximating $p(w_i|x_0^T, \theta)$ directly we define in the following section a network that is supposed to model the likelihood of the feature vector sequence $p(x_0^T|w_i, \theta)$.

### 3.2 THE MS-TDNN ARCHITECTURE

In Figure 2 the basic MS-TDNN architecture for handwriting recognition is shown. The first three layers constitute a standard TDNN with sliding input windows in each layer. In the current implementation of the system, a TDNN with 15 input

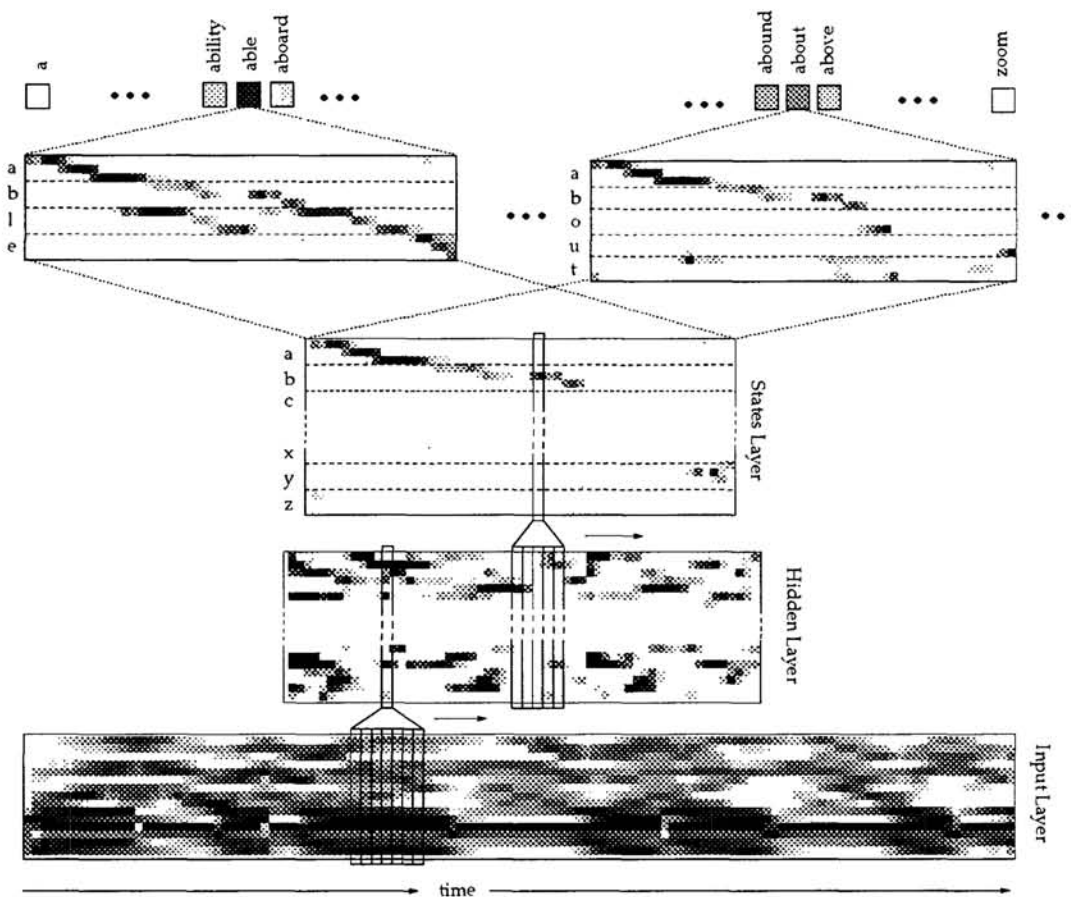

Figure 2: The Multi-State TDNN architecture, consisting of a 3-layer TDNN to estimate the a posteriori probabilities of the character states combined with word units, whose scores are derived from the word models by a Viterbi approximation of the likelihoods $p(\boldsymbol{x}_0^T | w_i)$.

units, 40 units in the hidden layer, and 78 state output units is used. There are 7 time delays both in the input and hidden layer.

The softmax normalized output of the states layer is interpreted as an estimate of the probabilities of the states $q_j$ given the input window $\boldsymbol{x}_{t-d}^{t+d} = \boldsymbol{x}_{t-d} \ldots \boldsymbol{x}_{t+d}$ for each time frame $t$, i.e.

$$p(q_j | \boldsymbol{x}_{t-d}^{t+d}) \quad \approx \quad \frac{\exp(\eta_j(t))}{\sum_k \exp(\eta_k(t))} \tag{1}$$

where $\eta_j(t)$ represents the weighted sum of inputs to state unit $j$ at time $t$. Based on these estimates, the output of the word units is defined to be a Viterbi approximation of the log likelihoods of the feature vector sequence given the word model

$w_i$, i.e.

$$\log p(\boldsymbol{x}_0^T|w_i) \approx \max_{q_0^T} \sum_{t=1}^{T} \log p(\boldsymbol{x}_{t-d}^{t+d}|q_t, w_i) + \log p(q_t|q_{t-1}, w_i)$$

$$\approx \max_{q_0^T} \sum_{t=1}^{T} \log p(q_t|\boldsymbol{x}_{t-d}^{t+d}) - \log p(q_t) + \log p(q_t|q_{t-1}, w_i). \quad (2)$$

Here, the maximum is over all possible sequences of states $q_0^T = q_0 \ldots q_T$ given a word model, $p(q_t|\boldsymbol{x}_{t-d}^{t+d})$ refers to the output of the states layer as defined in (1) and $p(q_t)$ is the prior probability of observing a state $q_t$ estimated on the training data.

## 3.3 TRAINING OF THE RECOGNIZER

During training the goal is to determine a set of parameters $\theta$ that will maximize the posterior probability $p(w|\boldsymbol{x}_0^T, \theta)$ for all training input sequences. But in order to make that maximization computationally feasible even for a large vocabulary system we had to simplify that maximum a posteriori approach to a maximum likelihood training procedure that maximizes $p(\boldsymbol{x}_0^T|w, \theta)$ for all words instead.

The first step of our maximum likelihood training is to bootstrap the recognizer using a subset of approximately 2,000 words of the training set that were labeled manually with the character boundaries to adjust the paths in the word layer correctly. After training on this hand-labeled data, the recognizer is used to label another larger set of unlabeled training data. Each pattern in this training set is processed by the recognizer. The boundaries determined automatically by the Viterbi alignment in the target word unit serve as new labels for this pattern. Then, in the second phase, the recognizer is retrained on both data sets to achieve the final performance of the recognizer.

## 4 EXPERIMENTS AND RESULTS

We have tested our system both on writer dependent and writer independent tasks with vocabulary sizes ranging from 400 up to 20,000 words. The word recognition results are shown in Table 1. The scaling of the recognition rates with respect to the vocabulary size is plotted in Figure 3b.

Table 1: Writer dependent and independent recognition results

| Task | Vocabulary Size | Writer Dependent | | Writer Independent | |
|---|---|---|---|---|---|
| | | Test Patterns | Recognition Rate | Test Patterns | Recognition Rate |
| crt_400 | 400 | 800 | 98.6% | 800 | 92.9% |
| wsj_1,000 | 1,000 | 800 | 97.8% | - | - |
| wsj_7,000 | 7,000 | - | - | 2,500 | 89.3% |
| wsj_10,000 | 10,000 | 1,600 | 92.1% | 2,500 | 87.7% |
| wsj_20,000 | 20,000 | 1,600 | 88.9% | 2,500 | 84.1% |

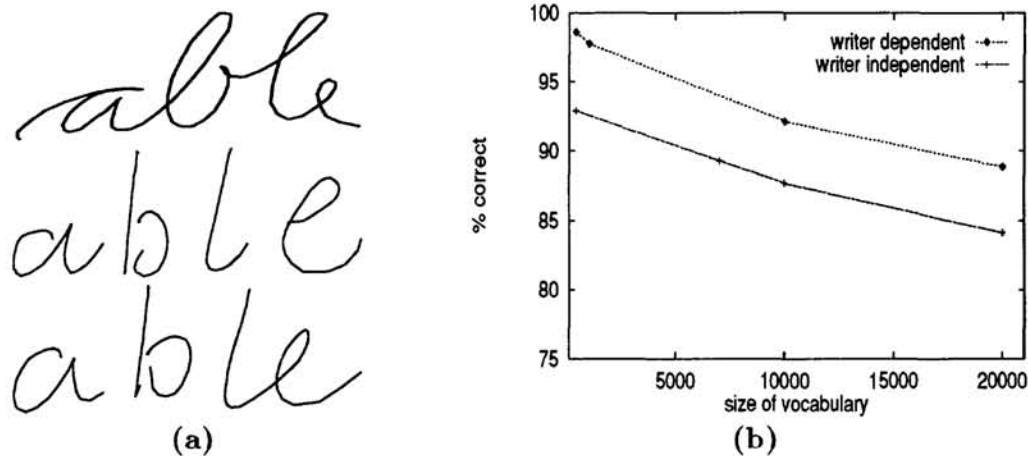

(a)                      (b)

Figure 3: (a) Different writing styles in the database: cursive (top), hand-printed (middle) and a mixture of both (bottom) (b) Recognition results with respect to the vocabulary size

For the writer dependent evaluation, the system was trained on 2,000 patterns from a 400 word vocabulary, written by a single writer, and tested on a disjunct set of patterns from the same writer. In the writer dependent case, the training set consisted of 4,000 patterns from a 7,000 word vocabulary, written by approximately 60 different writers. The test was performed on data from an independent set of 40 writers.

All data used in these experiments was collected at the University of Karlsruhe, Germany. Only minimal instructions were given to the writers. The writers were asked to write as natural as they would normally do on paper, without any restrictions in writing style. The consequence is, that the database is characterized by a high variety of different writing styles, ranging from hand-printed to strictly cursive patterns or a mixture of both writing styles (for example see Figure 3a). Additionally the native language of the writers was german, but the language of the dictionary is english. Therefore, frequent hesitations and corrections can be observed in the patterns of the database. But since this sort of input is typical for real world applications, a robust recognizer should be able to process these distorted patterns, too. From each of the writers a set of 50-100 isolated words, choosen randomly from the 7,000 word vocabulary, was collected.

The used vocabularies CRT (Conference Registration Task) and WSJ (ARPA Wall Street Journal Task) were originally defined for speech recognition evaluations. These vocabularies were chosen to take advantage of the synergy effects between handwriting recognition and speech recognition, since in our case the final goal is to integrate our speech recognizer JANUS [10] and the proposed **NPen$^{++}$** system into a multi-modal system.

## 5   CONCLUSIONS

In this paper we have presented the **NPen**$^{++}$ system, a neural recognizer for writer dependent and writer independent on-line cursive handwriting recognition. This system combines a robust input representation, which preserves the dynamic writing information, with a neural network integrating recognition and segmentation in a single framework. This architecture has been shown to be well suited for handling temporal sequences as provided by this kind of input.

Evaluation of the system on different tasks with vocabulary sizes ranging from 400 to 20,000 words has shown recognition rates from 92.9% to 84.1% in the writer independent case and from 98.6% to 88.9% in the writer dependent case. These results are especially promising because they were achieved with a small training set compared to other systems (e.g. [3]). As can be seen in Table 1, the system has proved to be virtually independent of the vocabulary. Though the system was trained on rather small vocabularies (e.g. 400 words in the writer dependent system), it generalizes well to completely different and much larger vocabularies.

## References

[1] S. Manke and U. Bodenhausen, "A Connectionist Recognizer for Cursive Handwriting Recognition", *Proceedings of the ICASSP-94*, Adelaide, April 1994.

[2] S. Manke, M. Finke, and A. Waibel, "Combining Bitmaps with Dynamic Writing Information for On-Line Handwriting Recognition", *Proceedings of the ICPR-94*, Jerusalem, October 1994.

[3] M. Schenkel, I. Guyon, and D. Henderson, "On-Line Cursive Script Recognition Using Time Delay Neural Networks and Hidden Markov Models", *Proceedings of the ICASSP-94*, Adelaide, April 1994.

[4] I. Guyon, P. Albrecht, Y. Le Cun, W. Denker, and W. Hubbard, "Design of a Neural Network Character Recognizer for a Touch Terminal", *Pattern Recognition*, 24(2), 1991.

[5] Y. Bengio and Y. LeCun. "Word Normalization for On-Line Handwritten Word Recognition", *Proceedings of the ICPR-94*, Jerusalem, October 1994.

[6] P. Haffner and A. Waibel, "Multi-State Time Delay Neural Networks for Continuous Speech Recognition", *Advances in Neural Information Processing Systems (NIPS-4)*, Morgan Kaufman, 1992.

[7] C. Bregler, H. Hild, S. Manke, and A. Waibel, "Improving Connected Letter Recognition by Lipreading", *Proceedings of the ICASSP-93*, Minneapolis, April 1993.

[8] A. Waibel, T. Hanazawa, G. Hinton, K. Shiano, and K. Lang, "Phoneme Recognition using Time-Delay Neural Networks", *IEEE Transactions on Acoustics, Speech and Signal Processing*, March 1989.

[9] W. Guerfali and R. Plamondon, "Normalizing and Restoring On-Line Handwriting", *Pattern Recognition*, 16(5), 1993.

[10] M. Woszczyna et al., "Janus 94: Towards Spontaneous Speech Translation", *Proceedings of the ICASSP-94*, Adelaide, April 1994.